# Inter-time segment information sharing for non-homogeneous dynamic Bayesian networks

**Dirk Husmeier & Frank Dondelinger**
Biomathematics & Statistics Scotland (BioSS)
JCMB, The King's Buildings, Edinburgh EH93JZ, United Kingdom
dirk@bioss.ac.uk, frank@bioss.ac.uk

**Sophie Lèbre**
Université de Strasbourg, LSIIT - UMR 7005, 67412 Illkirch, France
sophie.lebre@lsiit-cnrs.unistra.fr

## Abstract

Conventional dynamic Bayesian networks (DBNs) are based on the homogeneous Markov assumption, which is too restrictive in many practical applications. Various approaches to relax the homogeneity assumption have recently been proposed, allowing the network structure to change with time. However, unless time series are very long, this flexibility leads to the risk of overfitting and inflated inference uncertainty. In the present paper we investigate three regularization schemes based on inter-segment information sharing, choosing different prior distributions and different coupling schemes between nodes. We apply our method to gene expression time series obtained during the Drosophila life cycle, and compare the predicted segmentation with other state-of-the-art techniques. We conclude our evaluation with an application to synthetic biology, where the objective is to predict a known *in vivo* regulatory network of five genes in yeast.

## 1 Introduction

There is currently considerable interest in structure learning of dynamic Bayesian networks (DBNs), with a variety of applications in signal processing and computational biology; see e.g. [1, 2, 3]. The standard assumption underlying DBNs is that time-series have been generated from a homogeneous Markov process. This assumption is too restrictive in many applications and can potentially lead to erroneous conclusions. While there have been various efforts to relax the homogeneity assumption for undirected graphical models [4, 5], relaxing this restriction in DBNs is a more recent research topic [1, 2, 3, 6, 7, 8]. At present, none of the proposed methods is without its limitations, leaving room for further methodological innovation. The method proposed in [3, 8] is non-Bayesian. This requires certain regularization parameters to be optimized "externally", by applying information criteria (like AIC or BIC), cross-validation or bootstrapping. The first approach is suboptimal, the latter approaches are computationally expensive[1]. In the present paper we therefore follow the Bayesian paradigm, like [1, 2, 6, 7]. These approaches also have their limitations. The method proposed in [2] assumes a fixed network structure and only allows the interaction parameters to vary with time. This assumption is too rigid when looking at processes where changes in the overall regulatory network structure are expected, e.g. in morphogenesis or embryogenesis. The method proposed in [1] requires a discretization of the data, which incurs an inevitable information loss. These limitations are addressed in [6, 7], where the authors propose a method for continuous data that allows network structures associated with different nodes to change with time in different ways. However, this high flexibility causes potential problems when applied to time series with a low number of measurements, as typically available from systems biology, leading to overfitting or inflated

inference uncertainty. The objective of the work described in our paper is to propose a model that addresses the principled shortcomings of the three Bayesian methods mentioned above. Unlike [1], our model is continuous and therefore avoids the information loss inherent in a discretization of the data. Unlike [2], our model allows the network structure to change among segments, leading to greater model flexibility. As an improvement on [6, 7], our model introduces information sharing among time series segments, which provides an essential regularization effect.

## 2 Background: non-homogeneous DBNs without information coupling

This section summarizes briefly the non-homogeneous DBN proposed in [6, 7], which combines the Bayesian regression model of [10] with multiple changepoint processes and pursues Bayesian inference with reversible jump Markov chain Monte Carlo (RJMCMC) [11]. In what follows, we will refer to nodes as genes and to the network as a gene regulatory network. The method is not restricted to molecular systems biology, though.

### 2.1 Model

**Multiple changepoints:** Let $p$ be the number of observed genes, whose expression values $y = \{y_i(t)\}_{1 \leq i \leq p, 1 \leq t \leq N}$ are measured at $N$ time points. $\mathcal{M}$ represents a directed graph, i.e. the network defined by a set of directed edges among the $p$ genes. $\mathcal{M}_i$ is the subnetwork associated with target gene $i$, determined by the set of its parents (nodes with a directed edge feeding into gene $i$). The regulatory relationships among the genes, defined by $\mathcal{M}$, may vary across time, which we model with a multiple changepoint process. For each target gene $i$, an unknown number $k_i$ of changepoints define $k_i + 1$ non-overlapping segments. Segment $h = 1, .., k_i + 1$ starts at changepoint $\xi_i^{h-1}$ and stops before $\xi_i^h$, where $\xi_i = (\xi_i^0, ..., \xi_i^{h-1}, \xi_i^h, ..., \xi_i^{k_i+1})$ with $\xi_i^{h-1} < \xi_i^h$. To delimit the bounds, $\xi_i^0 = 2$ and $\xi_i^{k_i+1} = N + 1$. Thus vector $\xi_i$ has length $|\xi_i| = k_i + 2$. The set of changepoints is denoted by $\xi = \{\xi_i\}_{1 \leq i \leq p}$. This changepoint process induces a partition of the time series, $y_i^h = (y_i(t))_{\xi_i^{h-1} \leq t < \xi_i^h}$, with different structures $\mathcal{M}_i^h$ associated with the different segments $h \in \{1, \ldots, k_i + 1\}$. Identifiability is satisfied by ordering the changepoints based on their position in the time series.

**Regression model:** For all genes $i$, the random variable $Y_i(t)$ refers to the expression of gene $i$ at time $t$. Within any segment $h$, the expression of gene $i$ depends on the $p$ gene expression values measured at the previous time point through a regression model defined by (a) a set of $s_i^h$ parents denoted by $\mathcal{M}_i^h = \{j_1, ..., j_{s_i^h}\} \subseteq \{1, \ldots, p\}$, $|\mathcal{M}_i^h| = s_i^h$, and (b) a set of parameters $((a_{ij}^h)_{j \in 0..p}, \sigma_i^h)$; $a_{ij}^h \in \mathbb{R}$, $\sigma_i^h > 0$. For all $j \neq 0$, $a_{ij}^h = 0$ if $j \notin \mathcal{M}_i^h$. For all genes $i$, for all time points $t$ in segment $h$ ($\xi_i^{h-1} \leq t < \xi_i^h$), the random variable $Y_i(t)$ depends on the $p$ variables $\{Y_j(t-1)\}_{1 \leq j \leq p}$ according to

$$Y_i(t) = a_{i0}^h + \sum_{j \in \mathcal{M}_i^h} a_{ij}^h Y_j(t-1) + \varepsilon_i(t) \tag{1}$$

where the noise $\varepsilon_i(t)$ is assumed to be Gaussian with mean 0 and variance $(\sigma_i^h)^2$, $\varepsilon_i(t) \sim N(0, (\sigma_i^h)^2)$. We define $a_i^h = (a_{ij}^h)_{j \in 0..p}$.

### 2.2 Prior

The $k_i + 1$ segments are delimited by $k_i$ changepoints, where $k_i$ is distributed a priori as a truncated Poisson random variable with mean $\lambda$ and maximum $\overline{k} = N - 2$: $P(k_i|\lambda) \propto \frac{\lambda^{k_i}}{k_i!} \mathbb{1}_{\{k_i \leq \overline{k}\}}$. Conditional on $k_i$ changepoints, the changepoint positions vector $\xi_i = (\xi_i^0, \xi_i^1, ..., \xi_i^{k_i+1})$ takes non-overlapping integer values, which we take to be uniformly distributed a priori. There are $(N-2)$ possible positions for the $k_i$ changepoints, thus vector $\xi_i$ has prior density $P(\xi_i|k_i) = 1/\binom{N-2}{k_i}$. For all genes $i$ and all segments $h$, the number $s_i^h$ of parents for node $i$ follows a truncated Poisson distribution[2] with mean $\Lambda$ and maximum $\overline{s} = 5$: $P(s_i^h|\Lambda) \propto \frac{\Lambda^{s_i^h}}{s_i^h!} \mathbb{1}_{\{s_i^h \leq \overline{s}\}}$. Conditional on $s_i^h$, the prior for the parent set $\mathcal{M}_i^h$ is a uniform distribution over all parent sets with cardinality $s_i^h$: $P(\mathcal{M}_i^h | |\mathcal{M}_i^h| = s_i^h) = 1/\binom{p}{s_i^h}$. The overall prior on the network structures is given by marginalization:

$$P(\mathcal{M}_i^h|\Lambda) = \sum_{s_i^h=1}^{\overline{s}} P(\mathcal{M}_i^h|s_i^h) P(s_i^h|\Lambda) \tag{2}$$

Conditional on the parent set $\mathcal{M}_i^h$ of size $s_i^h$, the $s_i^h + 1$ regression coefficients, denoted by $a_{\mathcal{M}_i^h} = (a_{i0}^h, (a_{ij}^h)_{j \in \mathcal{M}_i^h})$, are assumed zero-mean multivariate Gaussian with covariance matrix $(\sigma_i^h)^2 \Sigma_{\mathcal{M}_i^h}$,

$$P(a_i^h | \mathcal{M}_i^h, \sigma_i^h) = |2\pi(\sigma_i^h)^2 \Sigma_{\mathcal{M}_i^h}|^{-\frac{1}{2}} \exp\left( -\frac{a_{\mathcal{M}_i^h}^\dagger \Sigma_{\mathcal{M}_i^h}^{-1} a_{\mathcal{M}_i^h}}{2(\sigma_i^h)^2} \right) \qquad (3)$$

where the symbol $\dagger$ denotes matrix transposition, $\Sigma_{\mathcal{M}_i^h} = \delta^{-2} D_{\mathcal{M}_i^h}^\dagger(y) D_{\mathcal{M}_i^h}(y)$ and $D_{\mathcal{M}_i^h}(y)$ is the $(\xi_i^h - \xi_i^{h-1}) \times (s_i^h + 1)$ matrix whose first column is a vector of 1 (for the constant in model (1)) and each $(j+1)^{th}$ column contains the observed values $(y_j(t))_{\xi_i^{h-1} - 1 \leq t < \xi_i^h - 1}$ for all factor gene $j$ in $\mathcal{M}_i^h$. This prior was also used in [10] and is motivated in [12]. Finally, the conjugate prior for the variance $(\sigma_i^h)^2$ is the inverse gamma distribution, $P((\sigma_i^h)^2) = \mathcal{IG}(v_0, \gamma_0)$. Following [6, 7], we set the hyper-hyperparameters for shape, $v_0 = 0.5$, and scale, $\gamma_0 = 0.05$, to fixed values that give a vague distribution. The terms $\lambda$ and $\Lambda$ can be interpreted as the expected number of changepoints and parents, respectively, and $\delta^2$ is the expected signal-to-noise ratio. These hyperparameters are drawn from vague conjugate hyperpriors, which are in the (inverse) gamma distribution family: $P(\Lambda) = P(\lambda) = \mathcal{Ga}(0.5, 1)$ and $P(\delta^2) = \mathcal{IG}(2, 0.2)$.

### 2.3 Posterior

Equation (1) implies that

$$P(y_i^h | \xi_i^{h-1}, \xi_i^h, \mathcal{M}_i^h, a_i^h, \sigma_i^h) = \left( \sqrt{2\pi} \sigma_i^h \right)^{-(\xi_i^h - \xi_i^{h-1})} \exp\left( -\frac{(y_i^h - D_{\mathcal{M}_i^h}(y) a_{\mathcal{M}_i^h})^\dagger (y_i^h - D_{\mathcal{M}_i^h}(y) a_{\mathcal{M}_i^h})}{2(\sigma_i^h)^2} \right) \qquad (4)$$

From Bayes theorem, the posterior is given by the following equation, where all prior distributions have been defined above:

$$P(k, \xi, \mathcal{M}, a, \sigma, \lambda, \Lambda, \delta^2 | y) \propto P(\delta^2) P(\lambda) P(\Lambda) \prod_{i=1}^{p} P(k_i | \lambda) P(\xi_i | k_i) \prod_{h=1}^{k_i} P(\mathcal{M}_i^h | \Lambda) \qquad (5)$$
$$P([\sigma_i^h]^2) P(a_i^h | \mathcal{M}_i^h, [\sigma_i^h]^2, \delta^2) P(y_i^h | \xi_i^{h-1}, \xi_i^h, \mathcal{M}_i^h, a_i^h, [\sigma_i^h]^2)$$

### 2.4 Inference

An attractive feature of the chosen model is that the marginalization over the parameters $a$ and $\sigma$ in the posterior distribution of (5) is analytically tractable:

$$P(k, \xi, \mathcal{M}, \lambda, \Lambda, \delta^2 | y) = \int P(k, \xi, \mathcal{M}, a, \sigma, \lambda, \Lambda, \delta^2 | y) \, da \, d\sigma \qquad (6)$$

See [6, 10] for details and an explicit expression. The number of changepoints and their location, $k, \xi$, the network structure $\mathcal{M}$ and the hyperparameters $\lambda, \Lambda, \delta^2$ can be sampled from the posterior $P(k, \xi, \mathcal{M}, \lambda, \Lambda, \delta^2 | y)$ with RJMCMC [11]. A detailed description can be found in [6, 10].

## 3 Model improvement: information coupling between segments

Allowing the network structure to change between segments leads to a highly flexible model. However, this approach faces a conceptual and a practical problem. The *practical* problem is potential model over-flexibility. If subsequent changepoints are close together, network structures have to be inferred from short time series segments. This will almost inevitably lead to overfitting (in a maximum likelihood context) or inflated inference uncertainty (in a Bayesian context). The *conceptual* problem is the underlying assumption that structures associated with different segments are a priori independent. This is not realistic. For instance, for the evolution of a gene regulatory network during embryogenesis, we would assume that the network evolves gradually and that networks associated with adjacent time intervals are a priori similar.

To address these problems, we propose three methods of information sharing among time series segments, as illustrated in Figure 1. The first method is based on hard information coupling between the nodes, using the exponential distribution proposed in [13]. The second scheme is also based on hard information coupling, but uses a binomial distribution with conjugate Beta prior. The third scheme is based on the same distributional assumptions as the second scheme, but replaces the hard by a soft information coupling scheme.

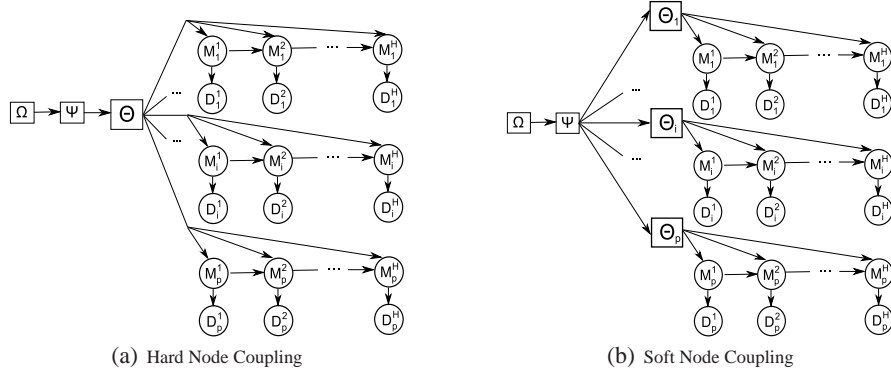

(a) Hard Node Coupling

(b) Soft Node Coupling

Figure 1: Hierarchical Bayesian models for inter-segment and inter-node information coupling. 1(a): Hard coupling between nodes with common hyperparameter $\Theta$ regulating the strength of the coupling between structures associated with adjacent segments, $\mathcal{M}_i^h$ and $\mathcal{M}_i^{h+1}$. This corresponds to the models in Section 3.1, with $\Theta = \beta$, $\Psi = [0, 10]$, and no $\Omega$, and Section 3.2, with $\Theta = \{a, b\}$, $\Psi = \{\alpha, \overline{\alpha}, \gamma, \overline{\gamma}\}$, and $\Omega = [0, 20]$. 1(b): Soft coupling between nodes, with node-specific hyperparameters $\Theta_i$ coupled via level2-hyperparameters $\Psi$. This corresponds to the model in Section 3.3, with $\Theta_i = \{a_i, b_i\}$, $\Psi = \{\alpha, \overline{\alpha}, \gamma, \overline{\gamma}\}$, and $\Omega = [0, 20]$.

## 3.1 Hard information coupling based on an exponential prior

Denote by $K_i := k_i + 1$ the total number of partitions in the time series associated with node $i$, and recall that each time series segment $y_i^h$ is associated with a separate subnetwork $\mathcal{M}_i^h$, $1 \leq h \leq K_i$. We impose a prior distribution $P(\mathcal{M}_i^h | \mathcal{M}_i^{h-1}, \beta)$ on the structures, and the joint probability distribution factorizes according to a Markovian dependence:

$$P(y_i^1, \ldots, y_i^{K_i}, \mathcal{M}_i^1, \ldots, \mathcal{M}_i^{K_i}, \beta) = \prod_{h=1}^{K_i} P(y_i^h | \mathcal{M}_i^h) P(\mathcal{M}_i^h | \mathcal{M}_i^{h-1}, \beta) P(\beta) \tag{7}$$

Similar to [13] we define

$$P(\mathcal{M}_i^h | \mathcal{M}_i^{h-1}, \beta) = \frac{\exp(-\beta |\mathcal{M}_i^h - \mathcal{M}_i^{h-1}|)}{Z_i(\beta, \mathcal{M}_i^{h-1})} \tag{8}$$

for $h \geq 2$, where $\beta$ is a hyperparameter that defines the strength of the coupling between $\mathcal{M}_i^h$ and $\mathcal{M}_i^{h-1}$, and $|.|$ denotes the Hamming distance. For $h = 1$, $P(\mathcal{M}_i^h)$ is given by (2). The denominator $Z(\beta, \mathcal{M}_i^{h-1})$ in (8) is a normalizing constant, also known as the partition function: $Z(\beta) = \sum_{\mathcal{M}_i^h \in \mathbb{M}} e^{-\beta |\mathcal{M}_i^h - \mathcal{M}_i^{h-1}|}$ where $\mathbb{M}$ is the set of all valid subnetwork structures. If we ignore any fan-in restriction that might have been imposed a priori (via $\overline{s}$), then the expression for the partition function can be simplified: $Z(\beta) \approx \prod_{j=1}^p Z_j(\beta)$, where $Z_j(\beta) = \sum_{e_j^h=0}^1 e^{-\beta |e_j^h - e_j^{h-1}|} = 1 + e^{-\beta}$ and hence $Z(\beta) = (1 + e^{-\beta})^p$. Inserting this expression into (8) gives:

$$P(\mathcal{M}_i^h | \mathcal{M}_i^{h-1}, \beta) = \frac{\exp(-\beta |\mathcal{M}_i^h - \mathcal{M}_i^{h-1}|)}{(1 + e^{-\beta})^p} \tag{9}$$

It is straightforward to integrate the proposed model into the RJMCMC scheme of [6, 7] as described in Section 2.4. When proposing a new network structure $\mathcal{M}_i^h \to \tilde{\mathcal{M}}_i^h$ for segment $h$, the prior probability ratio has to be replaced by: $\frac{P(\mathcal{M}_i^{h+1} | \tilde{\mathcal{M}}_i^h, \beta) P(\tilde{\mathcal{M}}_i^h | \mathcal{M}_i^{h-1}, \beta)}{P(\mathcal{M}_i^{h+1} | \mathcal{M}_i^h, \beta) P(\mathcal{M}_i^h | \mathcal{M}_i^{h-1}, \beta)}$. An additional MCMC step is introduced for sampling the hyperparameter $\beta$ from the posterior distribution. For a proposal move $\beta \to \tilde{\beta}$ with symmetric proposal probability $Q(\tilde{\beta}|\beta) = Q(\beta|\tilde{\beta})$ we get the following acceptance probability: $A(\tilde{\beta}|\beta) = \min\left\{\frac{P(\tilde{\beta})}{P(\beta)} \prod_{i=1}^p \prod_{h=2}^{K_i} \frac{\exp(-\tilde{\beta}|\mathcal{M}_i^h - \mathcal{M}_i^{h-1}|)}{\exp(-\beta|\mathcal{M}_i^h - \mathcal{M}_i^{h-1}|)} \frac{(1+e^{-\beta})^p}{(1+e^{-\tilde{\beta}})^p}, 1\right\}$ where in our study the hyperprior $P(\beta)$ was chosen as the uniform distribution on the interval $[0, 10]$.

## 3.2 Hard information coupling based on a binomial prior

An alternative way of information sharing among segments and nodes is by using a binomial prior:

$$P(\mathcal{M}_i^h | \mathcal{M}_i^{h-1}, a, b) = a^{N_1^1[h,i]} (1-a)^{N_1^0[h,i]} b^{N_0^0[h,i]} (1-b)^{N_0^1[h,i]} \tag{10}$$

where we have defined the following sufficient statistics: $N_1^1[h,i]$ is the number of edges in $\mathcal{M}_i^{h-1}$ that are matched by an edge in $\mathcal{M}_i^h$, $N_1^0[h,i]$ is the number of edges in $\mathcal{M}_i^{h-1}$ for which there is no edge in $\mathcal{M}_i^h$, $N_0^1[h,i]$ is the number of edges in $\mathcal{M}_i^h$ for which there is no edge in $\mathcal{M}_i^{h-1}$, and $N_0^0[h,i]$ is the number of coinciding non-edges in $\mathcal{M}_i^{h-1}$ and $\mathcal{M}_i^h$. Since the hyperparameters are shared, the joint distribution can be expressed as:

$$P(\{\mathcal{M}_i^h\}|a,b) \;=\; \prod_{i=1}^{p} P(\mathcal{M}_i^1) \prod_{h=1}^{K_i} P(\mathcal{M}_i^h|\mathcal{M}_i^{h-1},a,b) \;=\; a^{N_1^1}(1-a)^{N_1^0} b^{N_0^0}(1-b)^{N_0^1} \prod_{i=1}^{p} P(\mathcal{M}_i^1) \quad (11)$$

where we have defined $N_k^l = \sum_{i=1}^{p}\sum_{h=2}^{K_i} N_k^l[h,i]$, and the right-hand side follows from Eq. (10). The conjugate prior for the hyperparameters $a,b$ is a beta distribution, $P(a,b|\alpha,\overline{\alpha},\gamma,\overline{\gamma}) \propto a^{(\alpha-1)}(1-a)^{(\overline{\alpha}-1)}b^{(\gamma-1)}(1-b)^{(\overline{\gamma}-1)}$, which allows the hyperparameters to be integrated out in closed form:

$$P(\{\mathcal{M}_i^h\}|\alpha,\overline{\alpha},\gamma,\overline{\gamma}) \;=\; \int\int P(\{\mathcal{M}_i^h\}|a,b)P(a,b|\alpha,\overline{\alpha},\gamma,\overline{\gamma})\,da\,db \quad (12)$$

$$\propto\; \frac{\Gamma(\alpha+\overline{\alpha})}{\Gamma(\alpha)\Gamma(\overline{\alpha})}\frac{\Gamma(N_1^1+\alpha)\Gamma(N_1^0+\overline{\alpha})}{\Gamma(N_1^1+\alpha+N_1^0+\overline{\alpha})}\frac{\Gamma(\gamma+\overline{\gamma})}{\Gamma(\gamma)\Gamma(\overline{\gamma})}\frac{\Gamma(N_0^0+\gamma)\Gamma(N_0^1+\overline{\gamma})}{\Gamma(N_0^0+\gamma+N_0^1+\overline{\gamma})}$$

The level-2 hyperparameters $\alpha,\overline{\alpha},\gamma,\overline{\gamma}$ are given a uniform hyperprior over $[0,20]$. The MCMC scheme of Section 2.4 has to be modified as follows. When proposing a new network structure for node $i$ and segment $h$, $\mathcal{M}_i^h \to \tilde{\mathcal{M}}_i^h$, the structures $\mathcal{M}_i^h$ and $\tilde{\mathcal{M}}_i^h$ enter the prior probability ratio via the expression $P(\{\mathcal{M}_i^h\}|\alpha,\overline{\alpha},\gamma,\overline{\gamma})$, as $\frac{P(\{\mathcal{M}_i^1,...,\tilde{\mathcal{M}}_i^h,...,\mathcal{M}_i^{K_i}\}_{i=1}^{p}|\alpha,\overline{\alpha},\gamma,\overline{\gamma})}{P(\{\mathcal{M}_i^1,...,\mathcal{M}_i^h,...,\mathcal{M}_i^{K_i}\}_{i=1}^{p}|\alpha,\overline{\alpha},\gamma,\overline{\gamma})}$. Note that as a consequence of integrating out the hyperparameters, all network structures become interdependent, and information about the structures is contained in the sufficient statistics $N_1^1,N_1^0,N_0^1,N_0^0$. A new proposal move for the level-2 hyperparameters is added to the existing RJMCMC scheme of Section 2.4. New values for the level-2 hyperparameters $x \in \{\alpha,\overline{\alpha},\gamma,\overline{\gamma}\}$ are proposed from a uniform distribution over a fixed interval. For a move $x \to \tilde{x}$, the acceptance probability is:

$A(\tilde{x}|x) = \min\left\{\frac{P(\{\mathcal{M}_i^1,...,\mathcal{M}_i^{K_i}\}_{i=1}^{p}|\tilde{x},\{\alpha,\overline{\alpha},\gamma,\overline{\gamma}\}\setminus\tilde{x})}{P(\{\mathcal{M}_i^1,...,\mathcal{M}_i^{K_i}\}_{i=1}^{p}|x,\{\alpha,\overline{\alpha},\gamma,\overline{\gamma}\}\setminus x)},1\right\}$ where $\{\alpha,\overline{\alpha},\gamma,\overline{\gamma}\}\setminus x$ corresponds to $\{\overline{\alpha},\gamma,\overline{\gamma}\}$ if $x$ designates hyperparameter $\alpha$, and similarly for $\overline{\alpha},\gamma,\overline{\gamma}$.

### 3.3 Soft information coupling based on a binomial prior

We can relax the information sharing scheme from a hard to a soft coupling by introducing node-specific hyperparameters $a_i,b_i$ that are softly coupled via a common level-2 hyperprior, $P(a_i,b_i|\alpha,\overline{\alpha},\gamma,\overline{\gamma}) \propto a_i^{(\alpha-1)}(1-a_i)^{(\overline{\alpha}-1)}b_i^{(\gamma-1)}(1-b_i)^{(\overline{\gamma}-1)}$, as illustrated in Figure 1(b):

$$P(\mathcal{M}_i^h|\mathcal{M}_i^{h-1},a_i,b_i) \;=\; (a_i)^{N_1^1[h,i]}(1-a_i)^{N_1^0[h,i]}(b_i)^{N_0^0[h,i]}(1-b_i)^{N_0^1[h,i]} \quad (13)$$

This leads to a straightforward modification of eq. (11) – replacing $a,b$ by $a_i,b_i$ – from which we get as an equivalent to (13), using the definition $N_k^l[i] = \sum_{h=2}^{K_i} N_k^l[h,i]$:

$$P(\mathcal{M}_i^1,...,\mathcal{M}_i^{K_i}|\alpha,\overline{\alpha},\gamma,\overline{\gamma}) \propto \frac{\Gamma(\alpha+\overline{\alpha})}{\Gamma(\alpha)\Gamma(\overline{\alpha})}\frac{\Gamma(N_1^1[i]+\alpha)\Gamma(N_1^0[i]+\overline{\alpha})}{\Gamma(N_1^1[i]+\alpha+N_1^0[i]+\overline{\alpha})}\frac{\Gamma(\gamma+\overline{\gamma})}{\Gamma(\gamma)\Gamma(\overline{\gamma})}\frac{\Gamma(N_0^0[i]+\gamma)\Gamma(N_0^1[i]+\overline{\gamma})}{\Gamma(N_0^0[i]+\gamma+N_0^1[i]+\overline{\gamma})} \quad (14)$$

As in Section 3.2, we extend the RJMCMC scheme from Section 2.4 so that when proposing a new network structure, $\mathcal{M}_i^h \to \tilde{\mathcal{M}}_i^h$, the acceptance probability has to be updated with the prior ratio: $\frac{P(\mathcal{M}_i^1,...,\tilde{\mathcal{M}}_i^h,...,\mathcal{M}_i^{K_i}|\alpha,\overline{\alpha},\gamma,\overline{\gamma})}{P(\mathcal{M}_i^1,...,\mathcal{M}_i^h,...,\mathcal{M}_i^{K_i}|\alpha,\overline{\alpha},\gamma,\overline{\gamma})}$. In addition, we have to add a new level-2 hyperparameter update move $x \to \tilde{x}$, where the prior and proposal probabilities are the same as in Section 3.2, and the acceptance probability becomes: $A(\tilde{x}|x) = \min\left\{\prod_{i=1}^{p}\frac{P(\mathcal{M}_i^1,...,\mathcal{M}_i^{K_i}|\tilde{x},\{\alpha,\overline{\alpha},\gamma,\overline{\gamma}\}\setminus\tilde{x})}{P(\mathcal{M}_i^1,...,\mathcal{M}_i^{K_i}|x,\{\alpha,\overline{\alpha},\gamma,\overline{\gamma}\}\setminus x)},1\right\}$.

## 4 Results

The methods described in this paper have been implemented in R, based on code from [6, 7]. Our program sets up an RJMCMC simulation to sample the network structure, the changepoints and the hyperparameters from the posterior distribution. As a convergence diagnostic we monitor the potential scale reduction factor (PSRF) [14], computed from the within-chain and between-chain variances of marginal edge posterior probabilities. Values of PSRF$\leq 1.1$ are usually taken as indication of sufficient convergence. In our simulations, we extended the burn-in phase until a value of

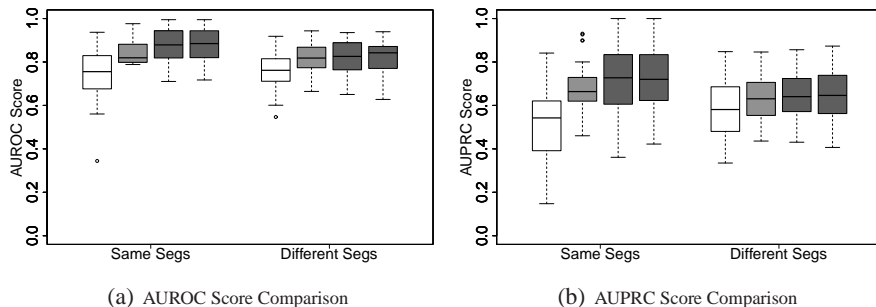

(a) AUROC Score Comparison    (b) AUPRC Score Comparison

Figure 2: Network reconstruction performance comparison of AUROC and AUPRC reconstruction scores for the four methods, HetDBN-0 (white), HetDBN-Exp (light grey), HetDBN-Bino1 (dark grey, left), HetDBN-Bino2 (dark grey, right). The boxplots show the distributions of the scores for 10 datasets with 4 network segments each, where the horizontal bar shows the median, the box margins show the 25th and 75th percentiles, the whiskers indicate data within 2 times the interquartile range, and circles are outliers. "Same Segs" means that all segments in a dataset have the same structure, while "Different Segs" indicates that structure changes are applied to the segments sequentially.

PSRF$\leq 1.05$ was reached, and then sampled 1000 network and changepoint configurations in intervals of 200 RJMCMC steps. From these samples we compute the marginal posterior probabilities of all potential interactions, which defines a ranking of the edges in the recovered network. When the true network is known, this allows us to construct the Receiver Operating Characteristic (ROC) curve (plotting the sensitivity or recall against the complementary specificity) and the precision-recall (PR) curve (plotting the precision against the recall), and to assess the network reconstruction accuracy in terms of the areas under these graphs (AUROC and AUPRC, respectively); see [15].

### 4.1 Comparative evaluation on simulated data

We randomly generated 10 networks with 10 nodes each, with the number of parents per node drawn from a Poisson distribution with mean $\lambda = 3$. To simulate changes in the network structure, we created 4 different network segments by drawing the number of changes from a Poisson distribution and applying the changes uniformly at random to edges and non-edges in the previous segment. For each segment, we generated a time series of length 15 using a linear regression model. The regression weights were drawn from a Gaussian $N(0,1)$, and Gaussian observation noise $N(0,1)$ was added. We compared the network reconstruction accuracy of the non-homogeneous DBN without information sharing proposed in [6, 7] (HetDBN-0) with the three information sharing approaches, based on the exponential prior from Section 3.1 (HetDBN-Exp), the binomial prior with hard node coupling from Section 3.2 (HetDBN-Bino1), and the binomial prior with soft node coupling from Section 3.3 (HetDBN-Bino2). Figures 2(a) and 2(b) shows the network reconstruction performance of the different information sharing methods in terms of AUROC and AUPRC scores. All information sharing methods show a clear improvement in network reconstruction over HetDBN-0, as confirmed by paired t-tests ($p < 0.01$). We investigated two different situations, the case where all segment structures are the same (although edge weights are allowed to vary) and the case where changes are applied sequentially to the segments[3]. Information sharing is most beneficial for the first case, but even when we introduce changes we still see an increase in the network reconstruction scores compared to HetDBN-0. When all segments are the same, HetDBN-Bino1 and HetDBN-Bino2 outperform HetDBN-Exp ($p < 0.05$), but there is no significant difference between the two binomial methods. Paired t-tests showed that all other differences in mean are significant. When the segments are different, all information sharing methods outperform HetDBN-0 ($p < 0.05$), but the difference between the information sharing methods is not significant.

### 4.2 Morphogenesis in *Drosophila melanogaster*

We applied our methods to a gene expression time series for eleven genes involved in the muscle development of *Drosophila melanogaster* [16]. The microarray data measured gene expression levels during all four major stages of morphogenesis: embryo, larva, pupa and adult. We investigated whether our methods were able to infer the correct changepoints corresponding to the known transitions between stages. Figure 3(a) shows the posterior probabilities of inferred changepoints for any gene using HetDBN-0, while Figure 3(c) shows the posterior probabilities for the information shar-

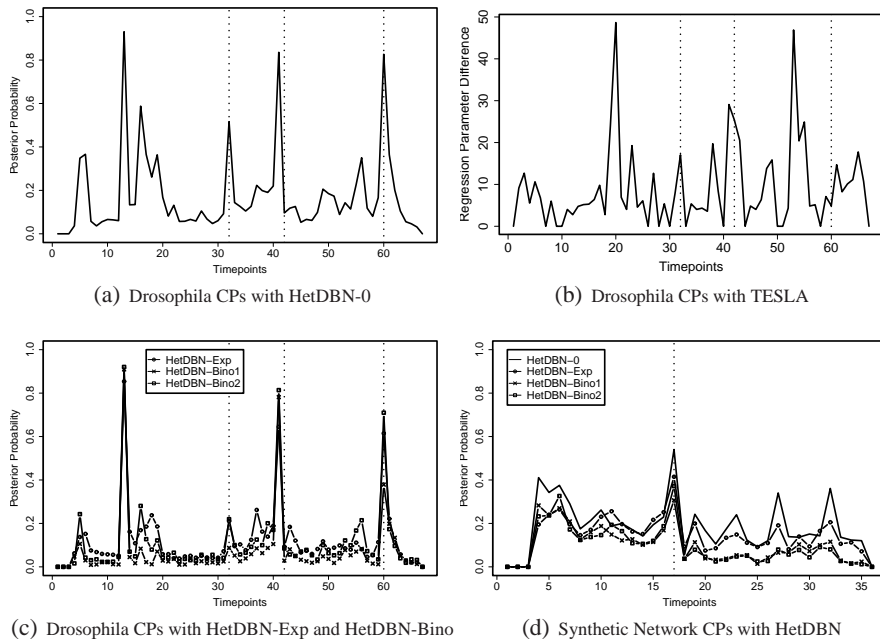

(a) Drosophila CPs with HetDBN-0

(b) Drosophila CPs with TESLA

(c) Drosophila CPs with HetDBN-Exp and HetDBN-Bino

(d) Synthetic Network CPs with HetDBN

Figure 3: Changepoints inferred on gene expression data related to morphogenesis in *Drosophila melanogaster*, and synthetic biology in *Saccharomyces cerevisiae* (yeast). All figures using HetDBN plot the posterior probability of a changepoint occurring for any node at a given time plotted against time. 3(a): HetDBN-0 changepoints for Drosophila (no information sharing) 3(b): TESLA, L1-norm of the difference of the regression parameter vectors associated with two adjacent time points plotted against time. 3(c): HetDBN changepoints for Drosophila with information sharing; the method is indicated by the legend. 3(d) HetDBN changepoints for the synthetic gene regulatory network in yeast. In 3(a)-3(c), the vertical dotted lines indicate the three morphogenic transitions, while in 3(d) the line indicates the boundary between "switch on" and "switch off" data.

ing methods. For comparison, we applied the method proposed in [3], using the authors' software package TESLA (Figure 3(b)). Robinson and Hartemink applied the discrete non-homogeneous DBN in [1] to the same data set, and a plot corresponding to Figure 3(b) can be found in their paper.

Our non-homogeneous DBN methods are generally more successful than TESLA, in that they recover changepoints for all three transitions (embryo → larva, larva → pupa, and pupa → adult). Figure 3(b) indicates that the last transition, pupa → adult, is less clearly detected with TESLA, and it is completely missing in [1]. Both our method as well as TESLA detect additional transitions during the embryo stage, which are missing in [1]. We would argue that a complex gene regulatory network is unlikely to transition into a new morphogenic phase all at once, and some pathways might have to undergo activational changes earlier in preparation for the morphogenic transition. As such, it is not implausible that additional transitions at the gene regulatory network level occur. However, a failure to detect known morphogenic transitions can clearly be seen as a shortcoming of a method, and on these grounds our model appears to outperform the two alternative ones. We note that the main effect of information sharing is to reduce the size of the smaller peaks, while keeping the three most salient peaks (corresponding to larva → pupa, and pupa → adult, and an extra transition in the embryo phase). This reflects the fact that these changepoints are associated with significant changes in network structure, and adds to the interpretability of the results. The drawback is that the third morphological transition (embryo → larva) is less pronounced.

### 4.3 Reconstruction of a synthetic gene regulatory network in *Saccharomyces cerevisiae*

The highly topical field of synthetic biology enables biologists to design known gene regulatory networks in living cells. In the work described in [17], a synthetic regulatory network of 5 genes was constructed in *Saccharomyces cerevisiae* (yeast), and gene expression time series were measured with RT-PCR for 16 and 21 time points under two experimental conditions, related to the carbon source: galactose ("switch on") and glucose ("switch off"). The authors tried to reconstruct the known gold-standard network from these time series with two established state-of-the-art methods from computational systems biology, one based on ordinary differential equations (ODEs), called

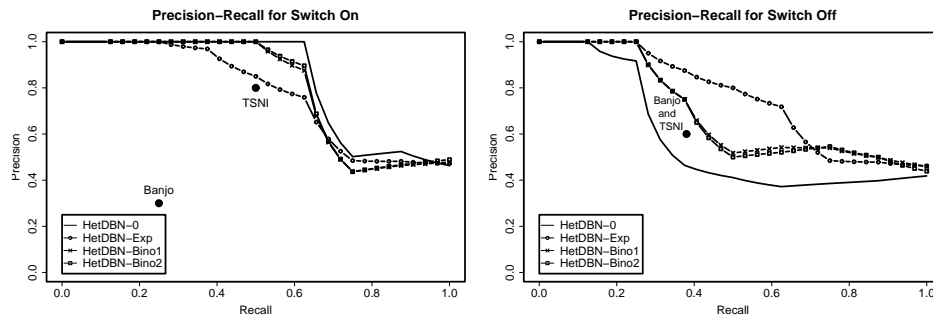

Figure 4: Reconstruction of a known gene regulatory network from synthetic biology in yeast. The network was reconstructed from two gene expression time series obtained with RT-PCR in two experimental conditions, reflecting the switch in the carbon source from galactose ("switch on") to glucose ("switch off"). The reconstruction accuracy of the methods proposed in Section 3, where the legend is explained, is shown in terms of precision (vertical axis) - recall (horizontal axis) curves. Results were averaged over 10 independent MCMC simulations. For comparison, fixed precision/recall scores are shown for two state-of-the-art methods reported in [17]: Banjo, a conventional DBN, and TSNI, a method based on ODEs.

TSNI, the other based on conventional DBNs, called Banjo; see [17] for details. Both methods are optimization-based and output a single network. By comparison with the known gold standard, the authors obtained the precision (proportion of predicted interactions that are correct) and recall (proportion of predicted true interactions) scores. In our study, we merged the time series from the two experimental conditions under exclusion of the boundary point[4], and applied the four non-homogeneous DBNs described before. Figure 3(d) shows the inferred marginal posterior probability of potential changepoints. The most significant changepoint is at the boundary between "switch on" and "switch off" data, confirming that the known true changepoint is consistently identified. The biological mechanism behind the other peaks is not known, and they are potentially spurious. Interestingly, the application of the proposed information-coupling schemes reduces the height of these peaks, with the binomial models having a stronger effect than the exponential one.

As we pursue a Bayesian inference scheme, we also obtain a ranking of the potential gene interactions in terms of their marginal posterior probabilities. From this we computed the precision-recall curves [15] shown in Figure 4. Our non-homogeneous DBNs with information sharing outperform Banjo and TSNI both in the "switch on" and the "switch off" phase. They also perform better than HetDBN-0 on the "switch off" data, but are slightly worse on the "switch on" data. Note that the reconstruction accuracy on the "switch off" data is generally poorer than on the "switch on" data [17]. Our results are thus plausible, suggesting that information sharing boosts the reconstruction accuracy on the poorer time series segment at the cost of a degraded performance on the stronger one. This effect is more pronounced for the exponential prior than for the binomial one, indicating a tighter coupling. The average areas under the PR curves, averaged over both phases ("switch on and off"), are as follows. HetDBN-0= 0.70, HetDBN-Exp= 0.77, HetDBN-Bino1= 0.75, HetDBN-Bino2= 0.75. Hence, the overall effect of information sharing is a performance improvement.

## 5  Conclusions

We have described a non-homogeneous DBN, which has various advantages over existing schemes: it does not require the data to be discretized (as opposed to [1]); it allows the network structure to change with time (as opposed to [2]); it includes three different regularization schemes based on inter-time segment information sharing (as opposed to [6, 7]); and it allows all hyperparameters to be inferred from the data via a consistent Bayesian inference scheme (as opposed to [3]). An evaluation on simulated data has demonstrated an improved performance over [6, 7] when information sharing is introduced. The application of our method to gene expression time series taken during the life cycle of *Drosophila melanogaster* has revealed better agreement with known morphogenic transitions than the methods of [1] and [3]. We have carried out a comparative evaluation of different information coupling schemes: a binomial versus an exponential prior, and hard versus soft coupling. In an application to data from a topical study in synthetic biology, our methods have outperformed two established network reconstruction methods from computational systems biology.

## Footnotes

[1]See [9] for a demonstration of the higher computational costs of bootstrapping over Bayesian approaches based on MCMC.

[2] A restrictive Poisson prior encourages sparsity of the network, and is therefore comparable to a sparse exponential prior, or an approach based on the LASSO.

[3]We chose to draw the number of changes from a Poisson with mean 1 for each node.

[4]When merging two time series $(x_1, \ldots, x_m)$ and $(y_1, \ldots, y_n)$, only the pairs $x_i \to x_j$ and $y_i \to y_j$ are presented to the DBN, while the pair $x_m \to y_1$ is excluded due to the obvious discontinuity.

# References

[1] J. W. Robinson and A. J. Hartemink. Non-stationary dynamic Bayesian networks. In D. Koller, D. Schuurmans, Y. Bengio, and L. Bottou, editors, *Advances in Neural Information Processing Systems (NIPS)*, volume 21, pages 1369–1376. Morgan Kaufmann Publishers, 2009.

[2] M. Grzegorczyk and D. Husmeier. Non-stationary continuous dynamic Bayesian networks. In Y. Bengio, D. Schuurmans, J. Lafferty, C. K. I. Williams, and A. Culotta, editors, *Advances in Neural Information Processing Systems (NIPS)*, volume 22, pages 682–690. 2009.

[3] A. Ahmed and E. P. Xing. Recovering time-varying networks of dependencies in social and biological studies. *Proceedings of the National Academy of Sciences*, 106:11878–11883, 2009.

[4] M. Talih and N. Hengartner. Structural learning with time-varying components: Tracking the cross-section of financial time series. *Journal of the Royal Statistical Society B*, 67(3):321–341, 2005.

[5] X. Xuan and K. Murphy. Modeling changing dependency structure in multivariate time series. In Zoubin Ghahramani, editor, *Proceedings of the 24th Annual International Conference on Machine Learning (ICML 2007)*, pages 1055–1062. Omnipress, 2007.

[6] S. Lèbre. *Stochastic process analysis for Genomics and Dynamic Bayesian Networks inference.* PhD thesis, Université d'Evry-Val-d'Essonne, France, 2007.

[7] S. Lèbre, J. Becq, F. Devaux, G. Lelandais, and M.P.H. Stumpf. Statistical inference of the time-varying structure of gene-regulation networks. *BMC Systems Biology*, 4(130), 2010.

[8] M. Kolar, L. Song, and E. Xing. Sparsistent learning of varying-coefficient models with structural changes. In Y. Bengio, D. Schuurmans, J. Lafferty, C. K. I. Williams, and A. Culotta, editors, *Advances in Neural Information Processing Systems (NIPS)*, volume 22, pages 1006–1014. 2009.

[9] B. Larget and D. L. Simon. Markov chain Monte Carlo algorithms for the Bayesian analysis of phylogenetic trees. *Molecular Biology and Evolution*, 16(6):750–759, 1999.

[10] C. Andrieu and A. Doucet. Joint Bayesian model selection and estimation of noisy sinusoids via reversible jump MCMC. *IEEE Transactions on Signal Processing*, 47(10):2667–2676, 1999.

[11] P. Green. Reversible jump Markov chain Monte Carlo computation and Bayesian model determination. *Biometrika*, 82:711–732, 1995.

[12] A. Zellner. On assessing prior distributions and Bayesian regression analysis with g-prior distributions. In P. Goel and A. Zellner, editors, *Bayesian Inference and Decision Techniques*, pages 233–243. Elsevier, 1986.

[13] A. V. Werhli and D. Husmeier. Gene regulatory network reconstruction by Bayesian integration of prior knowledge and/or different experimental conditions. *Journal of Bioinformatics and Computational Biology*, 6(3):543–572, 2008.

[14] A. Gelman and D.B. Rubin. Inference from iterative simulation using multiple sequences. *Statistical science*, 7(4):457–472, 1992.

[15] J. Davis and M. Goadrich. The relationship between precision-recall and ROC curves. In *Proceedings of the 23rd international conference on Machine Learning*, page 240. ACM, 2006.

[16] M.N. Arbeitman, E.E.M. Furlong, F. Imam, E. Johnson, B.H. Null, B.S. Baker, M.A. Krasnow, M.P. Scott, R.W. Davis, and K.P. White. Gene expression during the life cycle of Drosophila melanogaster. *Science*, 297(5590):2270–2275, 2002.

[17] I. Cantone, L. Marucci, F. Iorio, M. A Ricci, V. Belcastro, M. Bansal, S. Santini, M. di Bernardo, D. di Bernardo, and M. P Cosma. A yeast synthetic network for in vivo assessment of reverse-engineering and modeling approaches. *Cell*, 137(1):172181, 2009.

